# Is Early Vision Optimized for Extracting Higher-order Dependencies?

**Yan Karklin**
`yan+@cs.cmu.edu`

**Michael S. Lewicki**[*]
`lewicki@cnbc.cmu.edu`

Computer Science Department &
Center for the Neural Basis of Cognition
Carnegie Mellon University

## Abstract

Linear implementations of the efficient coding hypothesis, such as independent component analysis (ICA) and sparse coding models, have provided functional explanations for properties of simple cells in V1 [1, 2]. These models, however, ignore the non-linear behavior of neurons and fail to match individual and population properties of neural receptive fields in subtle but important ways. Hierarchical models, including Gaussian Scale Mixtures [3, 4] and other generative statistical models [5, 6], can capture higher-order regularities in natural images and explain non-linear aspects of neural processing such as normalization and context effects [6, 7]. Previously, it had been assumed that the lower level representation is independent of the hierarchy, and had been fixed when training these models. Here we examine the optimal lower-level representations derived in the context of a hierarchical model and find that the resulting representations are strikingly different from those based on linear models. Unlike the the basis functions and filters learned by ICA or sparse coding, these functions individually more closely resemble simple cell receptive fields and collectively span a broad range of spatial scales. Our work unifies several related approaches and observations about natural image structure and suggests that hierarchical models might yield better representations of image structure throughout the hierarchy.

## 1 Introduction

Efficient coding hypothesis has been proposed as a guiding computational principle for the analysis of early visual system and motivates the search for good statistical models of natural images. Early work revealed that image statistics are highly non-Gaussian [8, 9], and models such as independent component analysis (ICA) and sparse coding have been developed to capture these statistics to form efficient representations of natural images. It has been suggested that these models explain the basic computational goal of early visual cortex, as evidenced by the similarity between the learned parameters and the measured receptive fields of simple cells in V1.

---

[*]To whom correspondence should be addressed

In fact, it is not clear exactly how well these methods predict the shapes of neural receptive fields. There has been no thorough characterization of ICA and sparse coding results for different datasets, pre-processing methods, and specific learning algorithms employed, although some of these factors clearly affect the resulting representation [10]. When ICA or sparse coding is applied to natural images, the resulting basis functions resemble Gabor functions [1, 2] — 2D sine waves modulated by Gaussian envelopes — which also accurately model the shapes of simple cell receptive fields [11]. Often, these results are visualized in a transformed space, by taking the logarithm of the pixel intensities, sphering (whitening) the image space, or filtering the images to flatten their spectrum. When analyzed in the *original* image space, the learned filters (the models' analogues of neural receptive fields) do not exhibit the multi-scale properties of the visual system, as they tend to cluster at high spatial frequencies [10, 12]. Neural receptive fields, on the other hand, span a broad range of spatial scales, and exhibit distributions of spatial phase and other parameters unmatched by ICA and SC results [13,14]. Therefore, as models of early visual processing, these models fail to predict accurately either the individual or the population properties of cortical visual neurons.

Linear efficient coding methods are also limited in the type of statistical structure they can capture. Applied to natural images, their coefficients contain significant residual dependencies that cannot be accounted for by the linear form of the models. Several solutions have been proposed, including multiplicative Gaussian Scale Mixtures [4] and generative hierarchical models [5, 6]. These models capture some of the observed dependencies; but their analysis so far has been focused on the higher-order structure learned by the model. Meanwhile, the lower-level representation is either chosen *a priori* [4] or adapted separately, in the absence of the hierarchy [6] or with a fixed hierarchical structure specified in advance [5].

Here we examine whether the optimal lower-level representation of natural images is different when trained in the context of such non-linear hierarchical models. We also illustrate how the model not only describes sparse marginal densities and magnitude dependencies, but captures a variety of joint density functions that are consistent with previous observations and theoretical conjectures. We show that learned lower-level representations are strikingly different from those learned by the linear models: they are more multi-scale, spanning a wide range of spatial scales and phases of the Gabor sinusoid relative to the Gaussian envelope. Finally, we place these results in the context of whitening, gain control, and non-linear neural processing.

## 2  Fully adaptable scale mixture model

A simple and scalable model for natural image patches is a linear factor model, in which the data $\mathbf{x}$ are assumed to be generated as a linear combination of basis functions with additive noise

$$\mathbf{x} = \mathbf{A}\mathbf{u} + \boldsymbol{\epsilon}. \tag{1}$$

Typically, the noise is assumed to be Gaussian with variance $\sigma_\epsilon^2$, thus

$$P(\mathbf{x}|\mathbf{A}, \mathbf{u}) \propto \exp\left(-\sum_i \frac{1}{2\sigma_\epsilon^2} |\mathbf{x} - \mathbf{A}\mathbf{u}|_i^2\right). \tag{2}$$

The coefficients $\mathbf{u}$ are assumed to be mutually independent, and often modeled with sparse distributions (e.g. Laplacian) that reflect the non-Gaussian statistics of natural scenes [8,9],

$$P(\mathbf{u}) = \prod_i P(u_i) \propto \exp(-\sum_i |u_i|). \tag{3}$$

We can then adapt the basis functions $\mathbf{A}$ to maximize the expected log-likelihood of the data $L = \langle \log P(\mathbf{x}|\mathbf{A}) \rangle$ over the data ensemble, thereby learning a compact, efficient representation of structure in natural images. This is the model underlying the sparse coding algorithm [2] and closely related to independent component analysis (ICA) [1].

An alternative to fixed sparse priors for $\mathbf{u}$ (3) is to use a Gaussian Scale Mixture (GSM) model [3]. In these models, each observed coefficient $u_i$ is modeled as a product of random Gaussian variable $y_i$ and a multiplier $\lambda_i$,

$$u_i = \sqrt{\lambda_i} y_i \tag{4}$$

Conditional on the value of the multiplier $\lambda_i$, the probability $P(u_i|\lambda_i)$ is Gaussian with variance $\lambda_i$, but the form of the marginal distribution

$$P(u_i) = \int \mathcal{N}(0, \lambda_i) P(\lambda_i) d\lambda_i \tag{5}$$

depends on the probability function of $\lambda_i$ and can assume a variety of shapes, including sparse heavy-tailed functions that fit the observed distributions of wavelet and ICA coefficients [4]. This type of model can also account for the observed dependencies among coefficients $\mathbf{u}$, for example, by expressing them as pair-wise dependencies among the multiplier variables $\boldsymbol{\lambda}$ [4, 15].

A more general model, proposed in [6, 16], employs a hierarchical prior for $P(\mathbf{u})$ with adapted parameters tuned to the global patterns in higher-order dependencies. Specifically, the logarithm of the variances of $P(\mathbf{u})$ is assumed to be a linear function of the higher-order random variables $\mathbf{v}$,

$$\log \boldsymbol{\sigma}_u^2 = \mathbf{B}\mathbf{v} \,. \tag{6}$$

Conditional on the higher-order variables, the joint distribution of coefficients is factorisable, as in GSM. In fact, if the conditional density $P(\mathbf{u}|\mathbf{v})$ is Gaussian, this *Hierarchical Scale Mixture* (HSM) is equivalent to a GSM model, with $\boldsymbol{\lambda} = \boldsymbol{\sigma}_u^2$ and $P(\mathbf{u}|\boldsymbol{\lambda}) = P(\mathbf{u}|\mathbf{v}) = \mathcal{N}(0, \exp(\mathbf{B}\mathbf{v}))$, with the added advantage of a more flexible representation of higher-order statistical regularities in $\mathbf{B}$. Whereas previous GSM models of natural images focused on modeling local relationships between coefficients of fixed linear transforms, this general hierarchical formulation is fully adaptable, allowing us to recover the optimal lower-level representation $\mathbf{A}$, as well as the higher-order components $\mathbf{B}$.

Parameter estimation in the HSM involves adapting model parameters $\mathbf{A}$ and $\mathbf{B}$ to maximize data log-likelihood $L = \langle \log P(\mathbf{x}|\mathbf{A}, \mathbf{B}) \rangle$. The gradient descent algorithm for the estimation of $\mathbf{B}$ has been previously described (see [6]). The optimal lower-level basis $\mathbf{A}$ is computed similarly to the sparse coding algorithm — the goal is to minimize reconstruction error of the inferred MAP estimate $\hat{\mathbf{u}}$. However, $\hat{\mathbf{u}}$ is estimated not with a fixed sparsifying prior, but with a concurrently adapted hierarchical prior. If we assume a Gaussian conditional density $P(\mathbf{u}|\mathbf{v})$ and a standard-Normal prior $P(\mathbf{v})$, the MAP estimates are computed as

$$\{\hat{\mathbf{u}}, \hat{\mathbf{v}}\} = \arg \min_{\mathbf{u}, \mathbf{v}} P(\mathbf{u}, \mathbf{v}|\mathbf{x}, \mathbf{A}, \mathbf{B}) \tag{7}$$

$$= \arg \min_{\mathbf{u}, \mathbf{v}} P(\mathbf{x}|\mathbf{A}, \mathbf{B}, \mathbf{u}, \mathbf{v}) P(\mathbf{u}|\mathbf{v}) P(\mathbf{v}) \tag{8}$$

$$= \arg \min_{\mathbf{u}, \mathbf{v}} \left( \frac{1}{2\sigma_\epsilon^2} \sum_i |\mathbf{x} - \mathbf{A}\mathbf{u}|_i^2 + \sum_j \left( \frac{[\mathbf{B}\mathbf{v}]_j}{2} + \frac{u_j^2}{2e^{[\mathbf{B}\mathbf{v}]_j}} \right) + \sum_k \frac{v_k^2}{2} \right) \,. \tag{9}$$

Marginalizing over the latent higher-order variables in the hierarchical models leads to sparse distributions similar to the Laplacian and other density functions assumed in ICA.

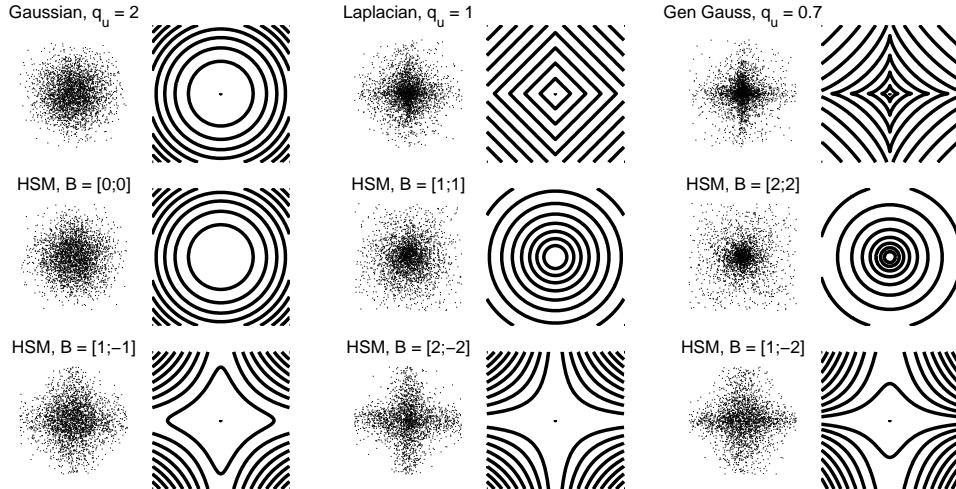

Figure 1: This model can describe a variety of joint density functions for coefficients $\mathbf{u}$. Here we show example scatter plots and contour plots of some bivariate densities. Top row: Gaussian, Laplacian, and generalized Gaussian densities of the form $p(u) \propto \exp(-|u|^q)$. Middle and bottom row: Hierarchical Scale Mixtures with different sets of parameters $\mathbf{B}$. For illustration, in the hierarchical models the dimensionality of $\mathbf{v}$ is 1, and the matrix $\mathbf{B}$ is simply a *column* vector. These densities are computed by marginalizing over the latent variables $\mathbf{v}$, here assumed to follow a standard normal distribution. Even with this simple hierarchy, the model can generate sparse star-shaped (bottom row) or radially symmetric (middle row) densities, as well as more complex non-symmetric densities (bottom right). In higher dimensions, it is possible to describe more complex joint distributions, with different marginals along different projections.

However, although the model distribution for individual coefficients is similar to the fixed sparse priors of ICA and sparse coding, the model is fundamentally non-linear and might yield a different lower-level representation; the coefficients $\mathbf{u}$ are no longer mutually independent, and the optimal set of basis functions must account for this.

Also, the shape of the *joint* marginal distribution in the space of all the coefficients is more complex than the i.i.d. joint density of the linear models. Bi-variate joint distributions of GSM coefficients can capture non-linear dependencies in wavelet coefficients [4]. In the fully adaptable HSM, however, the joint density can take a variety of shapes that depend on the learned parameters $\mathbf{B}$ (figure 1). Note that this model can produce sparse, star-shaped distributions as in the linear models, or radially symmetric distributions that cannot be described by the linear models. Such joint density profiles have been observed empirically in the responses of phase-offset wavelet coefficients to natural images and have inspired polar transformation and quadrature pair models [17] (as well as connections to phase-invariant neural responses). The model described here can capture these joint densities and others, but rather than assume this structure *a priori*, it learns it automatically from the data.

## 3   Methods

To examine how the lower-level representation is affected by the hierarchical model structure, we compared $\mathbf{A}$ learned by the sparse coding algorithm [2] and the HSM described above. The models were trained on $20 \times 20$ image patches sampled from 40 images of out-

door scenes in the Kyoto dataset [12]. We applied a low-pass radially symmetric filter to the full images to eliminate high corner frequencies (artifacts of the square sampling lattice), and removed the DC component from each image patch, but did no further pre-processing. All the results and analyses are reported in the original data space. Noise variance $\sigma_\epsilon^2$ was set to 0.1, and the basis functions were initialized to small random values and adapted on stochastically sampled batches of 300 patches. We ran the algorithm for 10,000 iterations with a step size of 0.1 (tapered for the last 1,000 iterations, once model parameters were relatively unchanging).

The parameters of the hierarchical model were estimated in a similar fashion. Gradient descent on $\mathbf{A}$ and $\mathbf{B}$ was performed in parallel using MAP estimates $\hat{\mathbf{u}}$ and $\hat{\mathbf{v}}$. The step size for adapting $\mathbf{B}$ was gradually increased from .0001 to .01, because emergence of the variance patterns requires some stabilization in the basis functions in $\mathbf{A}$.

Because encoding in the sparse coding and in the hierarchical model is a non-linear process, it is not possible to compare the inverse of $\mathbf{A}$ to physiological data. Instead, we estimated the corresponding *filters* using reverse correlation to derive a linear approximation to a non-linear system, which is also a common method for characterizing V1 simple cells. We analyzed the resulting filters by fitting them with 2D Gabor functions, then examining the distribution of their frequencies, phase, and orientation parameters.

## 4 Results

The shapes of basis functions and filters obtained with sparse coding have been previously analyzed and compared to neural receptive fields [10, 14]. However, some of the reported results were in the whitened space or obtained by training on filtered images. In the original space, sparse coding basis functions have very particular shapes: except for a few large, low frequency functions, all are localized, odd-symmetric, and span only a single period of the sinusoid (figure 2, top left). The estimated filters are similar but smaller (figure 2, bottom left), with peak spatial frequencies clustered at higher frequencies (figure 3).

In the hierarchical model, the learned representation is strikingly different (figure 2, right panels). Both the basis and the filters span a wider range of spatial scales, a result previously unobserved for models trained on non-preprocessed images, and one that is more consistent with physiological data [13, 14]. Also, the shapes of the basis functions are different — they more closely resemble Gabor functions, although they tend to be less smooth than the sparse coding basis functions. Both SC- and HSM-derived filters are well fit with Gabor functions.

We also compared the distributions of spatial phases for filters obtained with sparse coding and the hierarchical model (figure 4). While sparse coding filters exhibit a strong tendency for odd-symmetric phase profiles, the hierarchical model results in a much more uniform distribution of spatial phases. Although some phase asymmetry has been observed in simple cell receptive fields, their phase properties tend to be much more uniform than sparse coding filters [14].

In the hierarchical model, the higher-order representation $\mathbf{B}$ is also adapted to the statistical structure of natural images. Although the choice of the prior density for $\mathbf{v}$ (e.g. sparse or Gaussian) can determine the type of structure captured in $\mathbf{B}$, we discovered that it does not affect the nature of the lower-level representation. For the results reported here, we assumed a Gaussian prior on $\mathbf{v}$. Thus, as in other multi-variate Gaussian models, the precise directions of $\mathbf{B}$ are not important; the learned vectors only serve to collectively describe the volume of the space. In this case, they capture the *principal components of the log-variances*. Because we were interested specifically in the lower-level representation, we did not analyze the matrix $\mathbf{B}$ in detail, though the principal components of this space seem to

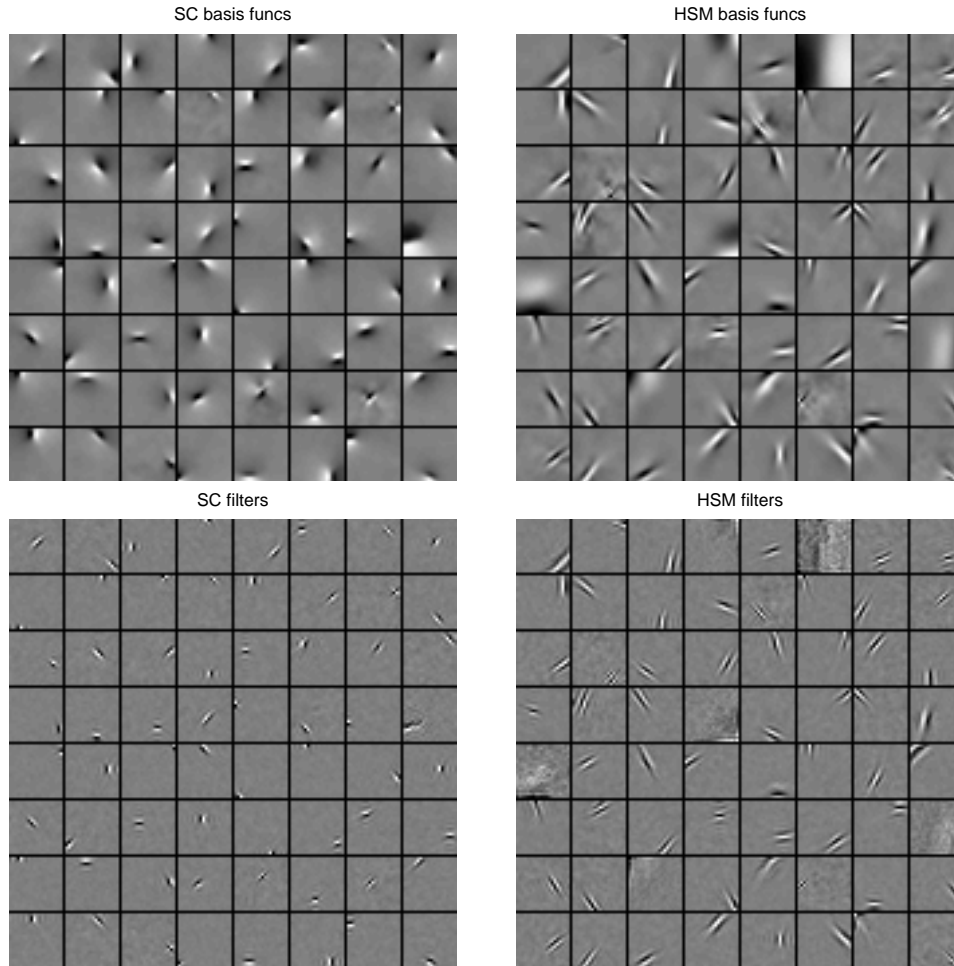

Figure 2: The lower-level representations learned by sparse coding (SC) and the hierarchical scale model (HSM). Shown are subsets of the learned basis functions and the estimates for the filters obtained with reverse correlation. These functions are displayed in the original image space.

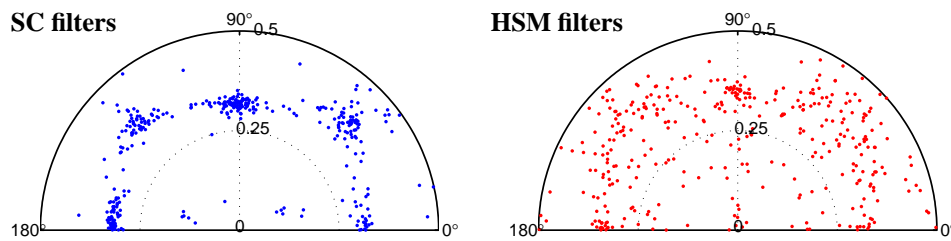

Figure 3: Scatter plots of peak frequencies and orientations of the Gabor functions fitted to the estimated filters. The units on the radial scale are cycles/pixel and the solid line is the Nyquist limit. Although both SC and HSM filters exhibit predominantly high spatial frequencies, the hierarchical model yields a representation that tiles the spatial frequency space much more evenly.

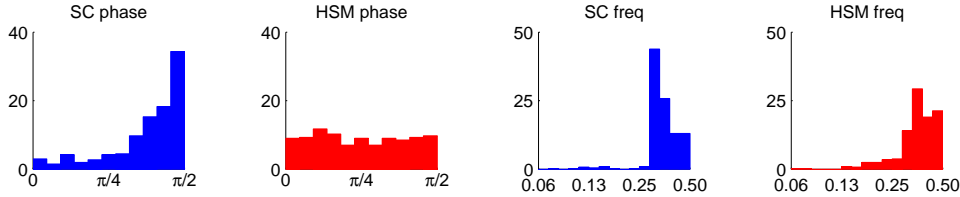

Figure 4: The distributions of phases and frequencies for Gabor functions fitted to sparse coding (SC) and hierarchical scale model (HSM) filters. The phase units specify the phase of the sinusoid in relation to the peak of the Gaussian envelope of the Gabor function; 0 is even-symmetric, $\pi/2$ is odd-symmetric. The frequency axes are in cycles/pixel.

group co-localized lower-level basis functions and separately represent spatial contrast and oriented image structure. As reported previously [6,16], with a sparse prior on $\mathbf{v}$, the model learns higher-order components that individually capture complex spatial, orientation, and scale regularities in image data.

## 5   Discussion

We have demonstrated that adapting a general hierarchical model yields lower-level representations that are significantly different than those obtained using fixed priors and linear generative models. The resulting basis functions and filters are multi-scale and more consistent with several observed characteristics of neural receptive fields.

It is interesting that the learned representations are similar to the results obtained when ICA or sparse coding is applied to whitened images (i.e. with a flattened power spectrum). This might be explained by the fact that whitening "spheres" the input space, normalizing the scale of different directions in the space. The hierarchical model is performing a similar scaling operation through the inference of higher-order variables $\mathbf{v}$ that scale the priors on basis function coefficients $\mathbf{u}$. Thus the model can rely on a generic "white" lower level representation, while employing an *adaptive* mechanism for normalizing the space, which accounts for non-stationary statistics on an image-by-image basis [6]. A related phenomenon in neural processing is gain control, which might be one specific type of a general adaptation process.

The flexibility of the hierarchical model allows us to learn a lower-level representation that is optimal in the context of the hierarchy. Thus, we expect the learned parameters to define a better statistical model for natural images than other approaches in which the lower-level representation or the higher-order dependencies are fixed in advance. For example, the flexible marginal distributions, illustrated in figure 1, should be able to capture a wider range of statistical structure in natural images. One way to quantify the benefit of an adapted lower-level representation is to apply the model to problems like image de-noising and filling-in missing pixels. Related models have achieved state-of-the-art performance [15, 18], and we are currently investigating whether the added flexibility of the model discussed here confers additional advantages.

Finally, although the results presented here are more consistent with the observed properties of neural receptive fields, several discrepancies remain. For example, our results, as well as those of other statistical models, fail to account for the prevalence of low spatial frequency receptive fields observed in V1. This could be a result of the specific choice of the distribution assumed by the model, although the described hierarchical framework makes few assumptions about the joint distribution of basis function coefficients. More likely, the non-stationary statistics of the natural scenes play a role in determining the properties of the learned representation. As suggested by previous results [10], different image data-sets

can lead to different parameters. This provides a strong motivation for training models with an "over-complete" basis, in which the number of basis functions is greater than the dimensionality of the input data [19]. In this case, different subsets of the basis functions can adapt to optimally represent different image contexts, and the population properties of such over-complete representations could be significantly different. It would be particularly interesting to investigate representations learned in these models in the context of a hierarchical model.

## References

[1] A. J. Bell and T. J. Sejnowski. The 'independent components' of natural scenes are edge filters. *Vision Research*, 37(23):3327–3338, 1997.

[2] B. A. Olshausen and D. J. Field. Emergence of simple-cell receptive-field properties by learning a sparse code for natural images. *Nature*, 381:607–609, 1996.

[3] D. F. Andrews and C. L. Mallows. Scale mixtures of normal distributions. *Journal of the Royal Statistical Society B*, 36(1):99–102, 1974.

[4] M. J. Wainwright, E. P. Simoncelli, and A. S. Willsky. Random cascades on wavelet trees and their use in analyzing and modeling natural images. *Applied Computational and Harmonic Analysis*, 11:89–123, 2001.

[5] A. Hyvärinen, P. O. Hoyer, and M. Inki. Topographic independent component analysis. *Neural Computation*, 13:1527–1558, 2001.

[6] Y. Karklin and M.S. Lewicki. A hierarchical bayesian model for learning non-linear statistical regularities in non-stationary natural signals. *Neural Computation*, 17:397–423, 2005.

[7] O. Schwartz and E. P. Simoncelli. Natural signal statistics and sensory gain control. *Nat. Neurosci.*, 4:819–825, 2001.

[8] D. Field. What is the goal of sensory coding. *Neural Computation*, 6:559–601, 1994.

[9] D. R. Ruderman and W. Bialek. Statistics of natural images: Scaling in the woods. *Physical Review Letters*, 73(6):814–818, 1994.

[10] J. H. van Hateren and A. van der Schaaf. Independent component filters of natural images compared with simple cells in primary visual cortex. *Proceedings of the Royal Society, London B*, 265:359–366, 1998.

[11] J. P. Jones and L. A. Palmer. An evaluation of the two-dimensional gabor filter model of simple receptive fields in cat striate cortex. *Journal of Neurophysiology*, 58(6):1233–1258, 1987.

[12] E. Doi and M. S. Lewicki. Sparse coding of natural images using an overcomplete set of limited capacity units. In *Advances in Neural Processing Information Systems 18*, 2004.

[13] R. L. De Valois, D. G. Albrecht, and L. G. Thorell. Spatial frequency selectivity of cells in macaque visual cortex. *Vision Research*, 22:545–559, 1982.

[14] D. L. Ringach. Spatial structure and symmetry of simple-cell receptive fields in macaque primary visual cortex. *Journal of Neurophysiology*, 88:455–463, 2002.

[15] J. Portilla, V. Strela, M. J. Wainwright, and E.P. Simoncelli. Image denoising using Gaussian scale mixtures in the wavelet domain. *IEEE Transactions on Image Processing*, 12:1338–1351, 2003.

[16] Y. Karklin and M.S. Lewicki. Learning higher-order structures in natural images. *Network: Computation in Neural Systems*, 14:483–499, 2003.

[17] C. Zetzsche and G. Krieger. Nonlinear neurons and highorder statistics: New approaches to human vision and electronic image processing. In B. Rogowitz and T.V. Pappas, editors, *Proc. SPIE on Human Vision and Electronic Imaging IV*, volume 3644, pages 2–33, 1999.

[18] M. S. Lewicki and B. A. Olshausen. A probabilistic framework for the adaptation and comparison of image codes. *Journal of the Optical Society of America A*, 16(7):1587–1601, 1999.

[19] B. A. Olshausen and D. J. Field. Sparse coding with an overcomplete basis set: A strategy employed by V1? *Vision Research*, 37(23), 1997.
